# Non-linear Metric Learning

**Dor Kedem, Stephen Tyree, Kilian Q. Weinberger**
Dept. of Comp. Sci. & Engi.
Washington U.
St. Louis, MO 63130
kedem.dor,swtyree,kilian@wustl.edu

**Fei Sha**
Dept. of Comp. Sci.
U. of Southern California
Los Angeles, CA 90089
feisha@usc.edu

**Gert Lanckriet**
Dept. of Elec. & Comp. Engineering
U. of California
La Jolla, CA 92093
gert@ece.ucsd.edu

## Abstract

In this paper, we introduce two novel metric learning algorithms, $\chi^2$-LMNN and GB-LMNN, which are explicitly designed to be *non-linear* and *easy-to-use*. The two approaches achieve this goal in fundamentally different ways: $\chi^2$-LMNN inherits the computational benefits of a linear mapping from linear metric learning, but uses a non-linear $\chi^2$-distance to explicitly capture similarities within histogram data sets; GB-LMNN applies gradient-boosting to learn non-linear mappings directly in function space and takes advantage of this approach's robustness, speed, parallelizability and insensitivity towards the single additional hyper-parameter. On various benchmark data sets, we demonstrate these methods not only match the current state-of-the-art in terms of kNN classification error, but in the case of $\chi^2$-LMNN, obtain best results in 19 out of 20 learning settings.

## 1   Introduction

How to compare examples is a fundamental question in machine learning. If an algorithm could perfectly determine whether two examples were semantically similar or dissimilar, most subsequent machine learning tasks would become trivial (*i.e.*, a nearest neighbor classifier will achieve perfect results). Guided by this motivation, a surge of recent research [10, 13, 15, 24, 31, 32] has focused on Mahalanobis metric learning. The resulting methods greatly improve the performance of metric dependent algorithms, such as k-means clustering and kNN classification, and have gained popularity in many research areas and applications within and beyond machine learning.

One reason for this success is the out-of-the-box usability and robustness of several popular methods to learn these linear metrics. So far, non-linear approaches [6, 18, 26, 30] to metric learning have not managed to replicate this success. Although more expressive, the optimization problems are often expensive to solve and plagued by sensitivity to many hyper-parameters. Ideally, we would like to develop easy-to-use black-box algorithms that learn new data representations for the use of established metrics. Further, non-linear transformations should be applied depending on the specifics of a given data set.

In this paper, we introduce two novel extensions to the popular Large Margin Nearest Neighbors (LMNN) framework [31] which provide *non-linear* capabilities and are applicable *out-of-the-box*. The two algorithms follow different approaches to achieve this goal:

(i) Our first algorithm, $\chi^2$-LMNN is specialized for histogram data. It generalizes the non-linear $\chi^2$-distance and learns a metric that strictly preserve the histogram properties of input data on a probability simplex. It successfully combines the simplicity and elegance of the LMNN objective and the domain-specific expressiveness of the $\chi^2$-distance.

(ii) Our second algorithm, gradient boosted LMNN (GB-LMNN) employs a non-linear mapping combined with a traditional Euclidean distance function. It is a natural extension of LMNN from linear to non-linear mappings. By training the non-linear transformation directly in function space with gradient-boosted regression trees (GBRT) [11] the resulting algorithm inherits the positive aspects of GBRT—its insensitivity to hyper-parameters, robustness against overfitting, speed and natural parallelizability [28].

Both approaches scale naturally to medium-sized data sets, can be optimized using standard techniques and only introduce a single additional hyper-parameter. We demonstrate the efficacy of both algorithms on several real-world data sets and observe two noticeable trends: i) GB-LMNN (with default settings) achieves state-of-the-art k-nearest neighbor classification errors with high consistency across all our data sets. For learning tasks where non-linearity is not required, it reduces to LMNN as a special case. On more complex data sets it reliably improves over linear metrics and matches or out-performs previous work on non-linear metric learning. ii) For data sampled from a simplex, $\chi^2$-LMNN is strongly superior to alternative approaches that do not explicitly incorporate the histogram aspect of the data—in fact it obtains best results in 19/20 learning settings.

## 2   Background and Notation

Let $\{(\mathbf{x}_1, y_1), \ldots, (\mathbf{x}_n, y_n)\} \in \mathcal{R}^d \times C$ be labeled training data with discrete labels $C = \{1, \ldots, c\}$. Large margin nearest neighbors (LMNN) [30, 31] is an algorithm to learn a Mahalanobis metric specifically to improve the classification error of k-nearest neighbors (kNN) [7] classification. As the kNN rule relies heavily on the underlying metric (a test input is classified by a majority vote amongst its $k$ nearest neighbors), it is a good indicator for the quality of the metric in use. The Mahalanobis metric can be viewed as a straight-forward generalization of the Euclidean metric,

$$\mathcal{D}_{\mathbf{L}}(\mathbf{x}_i, \mathbf{x}_j) = \|\mathbf{L}(\mathbf{x}_i - \mathbf{x}_j)\|_2, \tag{1}$$

parameterized by a matrix $\mathbf{L} \in \mathcal{R}^{d \times d}$, which in the case of LMNN is learned such that the linear transformation $\mathbf{x} \to \mathbf{Lx}$ better represents similarity in the target domain. In the remainder of this section we briefly review the necessary terminology and basic framework behind LMNN and refer the interested reader to [31] for more details.

**Local neighborhoods.** LMNN identifies two types of neighborhood relations between an input $\mathbf{x}_i$ and other inputs in the data set: For each $\mathbf{x}_i$, as a first step, $k$ dedicated *target neighbors* are identified prior to learning. These are the inputs that *should ideally* be the actual nearest neighbors after applying the transformation (we use the notation $j \rightsquigarrow i$ to indicate that $\mathbf{x}_j$ is a target neighbor of $\mathbf{x}_i$). A common heuristic for choosing target neighbors is picking the $k$ closest inputs (according to the Euclidean distance) to a given $\mathbf{x}_i$ within the same class. The second type of neighbors are *impostors*. These are inputs that *should not* be among the $k$-nearest neighbors — defined to be all inputs from a *different* class that are within the local neighborhood of $\mathbf{x}_i$.

**LMNN optimization.** The LMNN objective has two terms, one for each neighborhood objective: First, it reduces the distance between an instance and its target neighbors, thus pulling them closer and making the input's local neighborhood smaller. Second, it moves *impostor neighbors* (*i.e.*, differently labeled inputs) farther away so that the distances to impostors should exceed the distances to target neighbors by a large margin. Weinberger et. al [31] combine these two objectives into a single unconstrained optimization problem:

$$\min_{\mathbf{L}} \sum_{i,j:j \rightsquigarrow i} \underbrace{\mathcal{D}_{\mathbf{L}}(\mathbf{x}_i, \mathbf{x}_j)^2}_{pull \text{ target neighbor } \mathbf{x}_j \text{ closer}} + \mu \underbrace{\sum_{k \,:\, y_i \neq y_k} \left[ 1 + \mathcal{D}_{\mathbf{L}}(\mathbf{x}_i, \mathbf{x}_j)^2 - \mathcal{D}_{\mathbf{L}}(\mathbf{x}_i, \mathbf{x}_k)^2 \right]_+}_{push \text{ impostor } \mathbf{x}_k \text{ away, beyond target neighbor } \mathbf{x}_j \text{ by a large margin } \ell} \tag{2}$$

The parameter $\mu$ defines a trade-off between the two objectives and $[x]_+$ is defined as the hinge-loss $[x]_+ = \max(0, x)$. The optimization (2) can be transformed into a semidefinite program (SDP) [31] for which a global solution can be found efficiently. The large margin in (2) is set to 1 as its exact value only impacts the scale of $\mathbf{L}$ and not the kNN classifier.

**Dimensionality reduction.** As an extension to the original LMNN formulation, [26, 30] show that with $\mathbf{L} \in \mathcal{R}^{r \times d}$ with $r < d$, LMNN learns a projection into a lower-dimensional space $\mathcal{R}^r$ that still represents domain specific similarities. While this low-rank constraint breaks the convexity of the optimization problem, significant speed-ups [30] can be obtained when the kNN classifier is applied in the $r$-dimensional space — especially when combined with special-purpose data structures [33].

# 3   $\chi^2$-LMNN: Non-linear Distance Functions on the Probability Simplex

The original LMNN algorithm learns a linear transformation $\mathbf{L} \in \mathcal{R}^{d \times d}$ that captures semantic similarity for kNN classification on data in some Euclidean vector space $\mathcal{R}^d$. In this section we extend this formulation to settings in which data are sampled from a probability simplex $\mathcal{S}^d = \{\mathbf{x} \in \mathcal{R}^d | \mathbf{x} \geq 0, \mathbf{x}^\top \mathbf{1} = 1\}$, where $\mathbf{1} \in \mathcal{R}^d$ denotes the vector of all-ones. Each input $\mathbf{x}_i \in \mathcal{S}^d$ can be interpreted as a histogram over $d$ buckets. Such data are ubiquitous in computer vision where the histograms can be distributions over visual codebooks [27] or colors [25], in text-data as normalized bag-of-words or topic assignments [3], and many other fields [9, 17, 21].

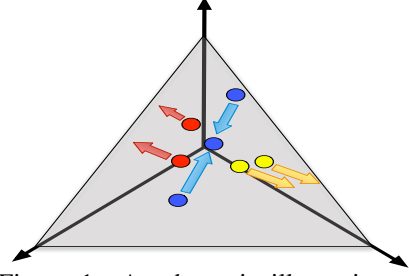

Figure 1: A schematic illustration of the $\chi^2$-LMNN optimization. The mapping is constrained to preserve all inputs on the simplex $\mathcal{S}^3$ (grey surface). The arrows indicate the push (red and yellow) and pull (blue) forces from the $\chi^2$-LMNN objective.

**Histogram distances.** The abundance of such data has sparked the development of several specialized distance metrics designed to compare histograms. Examples are Quadratic-Form distance [16], Earth Mover's Distance [21], Quadratic-Chi distance family [20] and $\chi^2$ histogram distance [16]. We focus explicitly on the latter. Transforming the inputs with a linear transformation learned with LMNN will almost certainly result in a loss of their histogram properties — and the ability to use such distances. In this section, we introduce our first non-linear extension for LMNN, to address this issue. In particular, we propose two significant changes to the original LMNN formulation: i) we learn a constrained mapping that keeps the transformed data on the simplex (illustrated in Figure 1), and ii) we optimize the kNN classification performance with respect to the non-linear $\chi^2$ histogram distance directly.

$\chi^2$ **histogram distance.** We focus on the $\chi^2$ histogram distance, whose origin is the $\chi^2$ statistical hypothesis test [19], and which has successfully been applied in many domains [8, 27, 29]. The $\chi^2$ distance is a bin-to-bin distance measurement, which takes into account the size of the bins and their differences. Formally, the $\chi^2$ distance is a well-defined metric $\chi^2 : \mathcal{S}^d \to [0, 1]$ defined as [20]

$$\chi^2(\mathbf{x}_i, \mathbf{x}_j) = \frac{1}{2} \sum_{f=1}^{d} \frac{([\mathbf{x}_i]_f - [\mathbf{x}_j]_f)^2}{[\mathbf{x}_i]_f + [\mathbf{x}_j]_f}, \tag{3}$$

where $[\mathbf{x}_i]_f$ indicates the $f^{th}$ feature value of the vector $\mathbf{x}_i$.

**Generalized $\chi^2$ distance.** First, analogous to the generalized Euclidean metric in (1), we generalize the $\chi^2$ distance with a linear transformation and introduce the pseudo-metric $\chi^2_{\mathbf{L}}(\mathbf{x}_i, \mathbf{x}_j)$, defined as

$$\chi^2_{\mathbf{L}}(\mathbf{x}_i, \mathbf{x}_j) = \chi^2(\mathbf{L}\mathbf{x}_i, \mathbf{L}\mathbf{x}_j). \tag{4}$$

The $\chi^2$ distance is only a well-defined metric within the simplex $\mathcal{S}^d$ and therefore we constrain $\mathbf{L}$ to map any $\mathbf{x}$ onto $\mathcal{S}^d$. We define the set of such simplex-preserving linear transformations as $\mathcal{P} = \{\mathbf{L} \in \mathcal{R}^{d \times d} : \forall \mathbf{x} \in \mathcal{S}^d, \mathbf{L}\mathbf{x} \in \mathcal{S}^d\}$.

$\chi^2$**-LMNN Objective.** To optimize the transformation $\mathbf{L}$ with respect to the $\chi^2$ histogram distance directly, we replace the Mahalanobis distance $\mathcal{D}_{\mathbf{L}}$ in (2) with $\chi^2_{\mathbf{L}}$ and obtain the following:

$$\min_{\mathbf{L} \in \mathcal{P}} \sum_{i,j: \ j \leadsto i} \chi^2_{\mathbf{L}}(\mathbf{x}_i, \mathbf{x}_j) + \mu \sum_{k: \ y_i \neq y_k} \left[ \ell + \chi^2_{\mathbf{L}}(\mathbf{x}_i, \mathbf{x}_j) - \chi^2_{\mathbf{L}}(\mathbf{x}_i, \mathbf{x}_k) \right]_+. \tag{5}$$

Besides the substituted distance function, there are two important changes in the optimization problem (5) compared to (2). First, as mentioned before, we have an additional constraint $\mathbf{L} \in \mathcal{P}$. Second, because (4) is not linear in $\mathbf{L}^\top \mathbf{L}$, different values for the margin parameter $\ell$ lead to truly different solutions (which differ not just up to a scaling factor as before). We therefore can no longer arbitrarily set $\ell = 1$. Instead, $\ell$ becomes an additional hyper-parameter of the model. We refer to this algorithm as $\chi^2$-LMNN.

**Optimization.** To learn (5), it can be shown $\mathbf{L} \in \mathcal{P}$ if and only if $\mathbf{L}$ is element-wise non-negative, *i.e.*, $\mathbf{L} \geq 0$, and each column is normalized, *i.e.*, $\sum_i L_{ij} = 1, \ \forall j$. These constraints are linear with respect

to $\mathbf{L}$ and we can optimize (5) efficiently with a projected sub-gradient method [2]. As an even faster optimization method, we propose a simple change of variables to generate an *unconstrained* version of (5). Let us define $f : \mathcal{R}^{d \times d} \to \mathcal{P}$ to be the column-wise soft-max operator

$$[f(\mathbf{A})]_{ij} = \frac{e^{A_{ij}}}{\sum_k e^{A_{kj}}}. \tag{6}$$

By design, all columns of $f(\mathbf{A})$ are normalized and every matrix entry is non-negative. The function $f(\cdot)$ is continuous and differentiable. By defining $\mathbf{L} = f(\mathbf{A})$ we obtain $\mathbf{L} \in \mathcal{P}$ for any choice of $\mathbf{A} \in \mathcal{R}^{d \times d}$. This allows us to minimize (5) with respect to $\mathbf{A}$ using *unconstrained* sub-gradient descent[1]. We initialize the optimization with $\mathbf{A} = 10\,\mathbf{I} + 0.01\,\mathbf{1}\mathbf{1}^\top$ (where $\mathbf{I}$ denotes the identity matrix) to approximate the non-transformed $\chi^2$ histogram distance after the change of variable ($f(\mathbf{A}) \approx \mathbf{I}$).

**Dimensionality Reduction.** Analogous to the original LMNN formulation (described in Section 2), we can restrict from a square matrix to $\mathbf{L} \in \mathcal{R}^{r \times d}$ with $r < d$. In this case $\chi^2$-LMNN learns a projection into a lower dimensional simplex $\mathbf{L} : \mathcal{S}^d \to \mathcal{S}^r$. All other parts of the algorithm change analogously. This extension can be very valuable to enable faster nearest neighbor search [33] especially for time-sensitive applications, *e.g.*, object recognition tasks in computer vision [27]. In section 6 we evaluate this version of $\chi^2$-LMNN under a range of settings for $r$.

## 4  GB-LMNN: Non-linear Transformations with Gradient Boosting

Whereas section 3 focuses on the learning scenario where a linear transformation is too general, in this section we explore the opposite case where it is too restrictive. Affine transformations preserve collinearity and ratios of distances along lines — *i.e.*, inputs on a straight line remain on a straight line and their relative distances are preserved. This can be too restrictive for data where similarities change locally (*e.g.*, because similar data lie on non-linear sub-manifolds). Chopra et al. [6] pioneered non-linear metric learning, using convolutional neural networks to learn embeddings for face-verification tasks. Inspired by their work, we propose to optimize the LMNN objective (2) directly in function space with gradient boosted CART trees [11]. Combining the learned transformation $\phi(\mathbf{x}) : \mathcal{R}^d \to \mathcal{R}^d$ with a Euclidean distance function has the capability to capture highly non-linear similarity relations. It can be optimized using standard techniques, naturally scales to large data sets while only introducing a single additional hyper-parameter in comparison with LMNN.

**Generalized LMNN.** To generalize the LMNN objective 2 to a non-linear transformation $\phi(\cdot)$, we denote the Euclidean distance after the transformation as

$$\mathcal{D}_\phi(\mathbf{x}_i, \mathbf{x}_j) = \|\phi(\mathbf{x}_i) - \phi(\mathbf{x}_j)\|_2, \tag{7}$$

which satisfies all properties of a well-defined pseudo-metric in the original input space. To optimize the LMNN objective directly with respect to $\mathcal{D}_\phi$, we follow the same steps as in Section 3 and substitute $\mathcal{D}_\phi$ for $\mathcal{D}_L$ in (2). The resulting unconstrained loss function becomes

$$\mathcal{L}(\phi) = \sum_{i,j:\, j \rightsquigarrow i} \|\phi(\mathbf{x}_i) - \phi(\mathbf{x}_j)\|_2^2 + \mu \sum_{k:\, y_i \neq y_k} \big[\, 1 + \|\phi(\mathbf{x}_i) - \phi(\mathbf{x}_j)\|_2^2 - \|\phi(\mathbf{x}_i) - \phi(\mathbf{x}_k)\|_2^2 \,\big]_+. \tag{8}$$

In its most general form, with an unspecified mapping $\phi$, (8) unifies most of the existing variations of LMNN metric learning. The original linear LMNN mapping [31] is a special case where $\phi(\mathbf{x}) = \mathbf{L}\mathbf{x}$. Kernelized versions [5, 12, 26] are captured by $\phi(\mathbf{x}) = \mathbf{L}\psi(\mathbf{x})$, producing the kernel $K(\mathbf{x}_i, \mathbf{x}_j) = \phi(\mathbf{x}_i)^\top \phi(\mathbf{x}_j) = \psi(\mathbf{x}_i)^\top \mathbf{L}^\top \mathbf{L}\psi(\mathbf{x}_j)$. The embedding of Globerson and Roweis [14] corresponds to the most expressive mapping function $\phi(\mathbf{x}_i) = \mathbf{z}_i$, where each input $\mathbf{x}_i$ is transformed independently to a new location $\mathbf{z}_i$ to satisfy similarity constraints — without out-of-sample extensions.

**GB-LMNN.** The previous examples vary widely in expressiveness, scalability, and generalization, largely as a consequence of the mapping function $\phi$. It is important to find the right non-linear form for $\phi$, and we believe an elegant solution lies in gradient boosted regression trees.

Our method, termed GB-LMNN, learns a global non-linear mapping. The construction of the mapping, an ensemble of multivariate regression trees selected by gradient boosting [11], minimizes the general LMNN objective (8) directly in function space. Formally, the GB-LMNN transformation

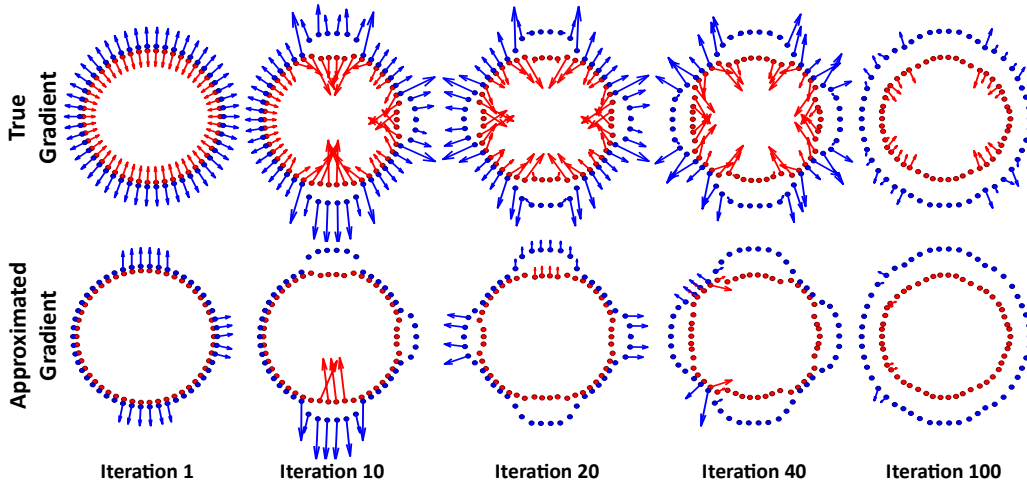

Figure 2: GB-LMNN illustrated on a toy data set sampled from two concentric circles of different classes (blue and red dots). The figure depicts the true gradient (*top row*) with respect to each input and its least squares approximation (*bottom row*) with a multi-variate regression tree (depth, $p = 4$).

is an additive function $\phi = \phi_0 + \alpha \sum_{t=1}^T h_t$ initialized by $\phi_0$ and constructed by iteratively adding regression trees $h_t$ of limited depth $p$ [4], each weighted by a learning rate $\alpha$. Individually, the trees are weak learners and are capable of learning only simple functions, but additively they form powerful ensembles with good generalization to out-of-sample data. In iteration $t$, the tree $h_t$ is selected greedily to best minimize the objective upon its addition to the ensemble,

$$\phi_t(\cdot) = \phi_{t-1}(\cdot) + \alpha h_t(\cdot), \quad \text{where } h_t \approx \underset{h \in \mathcal{T}^p}{\operatorname{argmin}} \mathcal{L}(\phi_{t-1} + \alpha h). \tag{9}$$

Here, $\mathcal{T}^p$ denotes the set of all regression trees of depth $p$. The (approximately) optimal tree $h_t$ is found by a first-order Taylor approximation of $\mathcal{L}$. This makes the optimization akin to a steepest descent step in function space, where $h_t$ is selected to approximate the negative gradient $g_t$ of the objective $\mathcal{L}(\phi_{t-1})$ with respect to the transformation learned at the previous iteration $\phi_{t-1}$. Since we learn an approximation of $g_t$ as a function of the training data, sub-gradients are computed with respect to each training input $\mathbf{x}_i$, and approximated by the tree $h_t(\cdot)$ in the least-squared sense,

$$h_t(\cdot) = \underset{h \in \mathcal{T}^p}{\operatorname{argmin}} \sum_{i=1}^n (g_t(\mathbf{x}_i) - h_t(\mathbf{x}_i))^2, \quad \text{where: } g_t(\mathbf{x}_i) = \frac{\partial \mathcal{L}(\phi_{t-1})}{\partial \phi_{t-1}(\mathbf{x}_i)}. \tag{10}$$

Intuitively, at each iteration, the tree $h_t(\cdot)$ of depth $p$ splits the input space into $2^p$ axis-aligned regions. All inputs that fall into one region are translated by a constant vector — consequently, the inputs in different regions are shifted in different directions. We learn the trees greedily with a modified version of the public-domain CART implementation pGBRT [28].

**Optimization details.** Since (8) is non-convex with respect to $\phi$, we initialize with the linear transformation learned by LMNN, $\phi_0 = \mathbf{L}\mathbf{x}$, making our method a non-linear refinement of LMNN. The only additional hyperparameter to the optimization is the maximum tree depth $p$ to which the algorithm is not particularly sensitive (we set $p = 6$). [2]

Figure 2 depicts a simple toy-example with concentric circles of inputs from two different classes. By design, the inputs are sampled such that the nearest neighbor for any given input is from the other class. A linear transformation is incapable of separating the two classes. However GB-LMNN produces a mapping with the desired separation. The figure illustrates the actual gradient (top row) and its approximation (bottom row). The limited-depth regression trees are unable to capture the gradient for all inputs in a single iteration. But by greedily focusing on inputs with the largest gradients or groups of inputs with the most easily encoded gradients, the gradient boosting process additively constructs the transformation function. At iteration 100, the gradients with respect to most inputs vanish, indicating that a local minimum of $\mathcal{L}(\phi)$ is almost reached — the inputs from the two classes are separated by a large margin.

**Dimensionality reduction.** Like linear LMNN and $\chi^2$-LMNN, it is possible to learn a non-linear transformation to a lower dimensional space, $\phi(\mathbf{x}) : \mathcal{R}^d \to \mathcal{R}^r$, $r \leq d$. Initialization is made with the rectangular matrix output of the dimensionality-reduced LMNN transformation, $\phi_0 = \mathbf{Lx}$ with $\mathbf{L} \in \mathcal{R}^{r \times d}$. Training proceeds by learning trees with $r$- rather than $d$-dimensional outputs.

## 5 Related Work

There have been previous attempts to generalize learning linear distances to nonlinear metrics. The nonlinear mapping $\phi(\mathbf{x})$ of eq. (7) can be implemented with kernels [5, 12, 18, 26]. These extensions have the advantages of maintaining computational tractability as convex optimization problems. However, their utility is limited inherently by the sizes of kernel matrices .Weinberger et. al [30] propose $M^2$-LMNN, a locally linear extension to LMNN. They partition the space into multiple regions, and jointly learn a separate metric for each region—however, these local metrics do *not* give rise to a global metric and distances between inputs within different regions are not well-defined.

Neural network-based approaches offer the flexibility of learning arbitrarily complex nonlinear mappings [6]. However, they often demand high computational expense, not only in parameter fitting but also in model selection and hyper-parameter tuning. Of particular relevance to our GB-LMNN work is the use of boosting ensembles to learn distances between bit-vectors [1, 23]. Note that their goals are to preserve distances computed by locality sensitive hashing to enable fast search and retrieval. Ours are very different: we alter the distances *discriminatively* to minimize classification error.

Our work on $\chi^2$-LMNN echoes the recent interest in learning earth-mover-distance (EMD) which is also frequently used in measuring similarities between histogram-type data [9]. Despite its name, EMD is not necessarily a metric [20]. Investigating the link between our work and those new advances is a subject for future work.

## 6 Experimental Results

We evaluate our non-linear metric learning algorithms against several competitive methods. The effectiveness of learned metrics is assessed by $k$NN classification error. Our open-source implementations are available for download at `http://www.cse.wustl.edu/~kilian/code/code.html`.

**GB-LMNN** We compare the non-linear global metric learned by GB-LMNN to three linear metrics: the Euclidean metric and metrics learned by LMNN [31] and Information-Theoretic Metric Learning (ITML) [10]. Both optimize similar discriminative loss functions. We also compare to the metrics learned by Multi-Metric LMNN ($M^2$-LMNN) [30]. $M^2$-LMNN learns $|C|$ linear metrics, one for each the input labels.

We evaluate these methods and our GB-LMNN on several medium-sized data sets: *ISOLET*, *USPS* and *Letters* from the UCI repository. ISOLET and USPS have predefined test sets, otherwise results are averaged over 5 train/test splits ($80\%/20\%$). A hold-out set of $25\%$ of the training set[3] is used to assign hyper-parameters and to determine feature pre-processing (*i.e.*, feature-wise normalization). We set $k = 3$ for $k$NN classification, following [31]. Table 1 reports the means and standard errors of each approach (standard error is omitted for data with pre-defined test sets), with numbers in bold font indicating the best results up to one standard error.

On all three datasets, GB-LMNN outperforms methods of learning linear metrics. This shows the benefit of learning nonlinear metrics. On *Letters*, GB-LMNN outperforms the second-best method $M^2$-LMNN by significant margins. On the other two, GB-LMNN is as good as $M^2$-LMNN.

We also apply GB-LMNN to four datasets with *histogram* data — setting the stage for an interesting comparison to $\chi^2$-LMNN below. The results are displayed on the right side of the table. These datasets are popularly used in computer vision for object recognition [22]. Data instances are 800-bin histograms of visual codebook entries. There are ten common categories to the four datasets and we use them for multiway classification with $k$NN.

Neither method evaluated so far is specifically adapted to histogram features. Especially linear models, such as LMNN and ITML, are expected to fumble over the intricate similarities that such

| | Euclidean space | | | probability simplex | | | |
|---|---|---|---|---|---|---|---|
| | isolet | usps | letters | dslr | webcam | amazon | caltech |
| | n=7797,d=172 | n=9298,d=256 | n=20000,d=16 | n=157,d=800 | n=295,d=800 | n=958,d=800 | n=1123,d=800 |
| Euclidean | 8.4 | 6.2 | 6.0±0.2 | 60.6±3.1 | 43.8±1.7 | 33.7±0.7 | 53.8±1.3 |
| ITML | 5.3±0.0 | 5.7 | 6.0±0.2 | *25.0±3.0* | *12.4±1.6* | 31.6±1.2 | 52.2±2.1 |
| LMNN | **1.5±0.1** | 2.6 | 3.8±0.3 | 28.9±1.6 | 15.8±3.0 | 31.8±1.4 | *50.9±1.4* |
| M$^2$-LMNN | **1.4±0.1** | **2.5** | 3.8±0.2 | 27.4±2.1 | 15.7±3.2 | *31.2±1.1* | 51.5±1.5 |
| GB-LMNN | **1.4±0.0** | **2.5** | **1.9±0.1** | *22.9±2.7* | *12.4±0.9* | *29.6±1.7* | *49.8±1.0* |
| $\chi^2$ | - | - | - | 22.2±1.8 | 13.0±1.2 | 34.3±1.0 | 58.8±1.1 |
| QCS | - | - | - | 25.6±2.7 | 19.4±1.1 | 33.9±2.0 | 57.2±1.2 |
| QCN | - | - | - | 27.8±4.1 | 17.5±2.1 | 34.5±1.5 | 56.1±1.2 |
| $\chi^2$-LMNN | - | - | - | **20.6±1.1** | **8.3±0.9** | **23.7±0.8** | **46.5±1.1** |

Table 1: $k$NN classification error (in %, ± standard error where applicable), for general methods (top section) and histogram methods (bottom section). Best results up to one standard error in **bold**. Best results among general methods for simplex data in *red italics*.

data types may encode. As shown in the table, GB-LMNN consistently outperforms the linear methods and $M^2$-LMNN.

$\chi^2$**-LMNN** In Table 1, we compare $\chi^2$-LMNN to other methods for computing distances on histogram features: $\chi^2$-distance *without* transformation (equivalent to our parameterized distance $\chi^2_{\mathbf{L}}$ distance with the transformation $\mathbf{L}$ being the identity matrix), Quadratic-Chi-Squared (QCS) and Quadratic-Chi-Normalized (QCN) distances, defined in [20]. For QCS and QCN, we use histogram intersection as the ground distance. Unlike our approach, none of these is *discriminatively* learned from data. $\chi^2$-LMNN outperforms all other methods significantly.

It is also instructive to compare the results to the performance of non-histogram specific methods. We observe that LMNN performs better than the standard $\chi^2$-distance on *Amazon* and *Caltech*. This seems to suggest that for those two datasets, linear metrics may be adequate and GB-LMNN's nonlinear mapping might not be able to provide extra expressiveness and benefits. This is confirmed in Table 1: GB-LMNN improves performance less significantly for *Amazon* and *Caltech* than for the other two datasets, *DSLR* and *Webcam*. For the latter two, on the contrary, LMNN performs worse than $\chi^2$-distance. In such cases, GB-LMNN's nonlinear mapping seems more beneficial. It provides a significant performance boost, and matches the performance of $\chi^2$-distance (up to one standard-error). Nonetheless, despite learning a nonlinear mapping, GB-LMNN still underperforms $\chi^2$-LMNN. In other words, it is possible that *no matter how flexible a nonlinear mapping could be, it is still best to use metrics that respect the semantic features of the data.*

**Dimensionality reduction.** GB-LMNN and $\chi^2$-LMNN are both capable of performing dimensionality reduction. We compare these with three dimensionality reduction methods (PCA, LMNN, and $M^2$-LMNN) on the histogram datasets and the larger UCI datasets. Each dataset is reduced to an output dimensionality of $r = 10, 20, 40, 80$ features. As we can see from the results in Table 6, it is fair to say that GB-LMNN performs comparably with LMNN and $M^2$-LMNN, whereas $\chi^2$-LMNN obtains at times phenomenally low $k$NN error rates on the histograms data sets (*e.g.*, *Webcam*). This suggests that dimensionality reduction of histogram data can be highly effective, if the data properties are carefully incorporated in the process. We do not apply dimensionality reduction to *Letters* as it already lies in a low-dimensional space ($d = 16$).

**Sensitivity to parameters.** One of the most compelling aspects of our methods is that each introduces only a single new hyper-parameter to the LMNN framework. During our experiments, $\ell$ was selected by cross-validation and $p$ was fixed to $p = 6$. We found very little sensitivity in GB-LMNN to regression tree depth, while large margin size was an important but well-behaved parameter for $\chi^2$-LMNN. Additional graphs are included in the supplementary material.

# 7 Conclusion and Future Work

In this paper we introduced two non-linear extensions to LMNN, $\chi^2$-LMNN and GB-LMNN. Although based on fundamentally different approaches, both algorithms lead to significant improvements over the original (linear) LMNN metrics and match or out-perform existing non-linear algorithms. The non-convexity of our proposed methods does not seem to impact their performance,

|  |  | Euclidean space | | probability simplex | | | |
|---|---|---|---|---|---|---|---|
|  |  | isolet | usps | dslr | webcam | amazon | caltech |
| r=10 | PCA | 26.6 | 10.1 | **42.8±3.7** | 32.7±1.6 | 49.1±2.2 | 63.8±1.1 |
|  | LMNN | 4.20±0.1 | 6.0 | 56.1±2.0 | 38.1±2.5 | 43.6±4.6 | **54.6±2.1** |
|  | M²-LMNN | 4.3±0.2 | **5.2** | 56.1±2.0 | 38.4±2.7 | 42.8±1.7 | **55.0±2.2** |
|  | GB-LMNN | **3.7±0.0** | 5.3 | 46.7±7.4 | 34.6±2.6 | 41.6±2.7 | **55.8±2.1** |
|  | $\chi^2$-LMNN | - | - | **43.3±2.6** | **17.8±2.9** | **33.9±1.1** | 54.7±1.9 |
| r=20 | PCA | 15.1 | 6.6 | 46.1±3.7 | 27.3±1.7 | 43.9±1.1 | 59.9±0.5 |
|  | LMNN | **2.1±0.1** | 3.8 | 53.3±2.8 | 34.0±2.9 | 39.9±1.5 | **55.4±1.7** |
|  | M²-LMNN | **2.1±0.2** | **3.3** | 53.3±2.8 | 34.3±2.6 | 40.3±1.3 | 55.5±1.5 |
|  | GB-LMNN | **2.0±0.1** | 3.8 | 50.0±3.4 | 33.0±2.8 | 38.7±0.8 | **53.7±1.3** |
|  | $\chi^2$-LMNN | - | - | **31.1±3.2** | **14.3±3.1** | **33.1±0.9** | 55.6±0.9 |
| r=40 | PCA | 11.0 | 6.0 | 46.7±3.0 | 29.2±2.2 | 43.1±1.6 | 57.7±0.5 |
|  | LMNN | 1.5±0.0 | 3.2 | 51.7±0.7 | 36.8±2.0 | 39.4±1.0 | 56.1±1.5 |
|  | M²-LMNN | **1.2±0.1** | 3.2 | 51.7±0.7 | 36.2±1.3 | 39.4±1.3 | 56.1±1.6 |
|  | GB-LMNN | 1.4±0.1 | **2.9** | 50.0±2.1 | 31.7±1.3 | 39.3±1.3 | **53.3±1.4** |
|  | $\chi^2$-LMNN | - | - | **33.3±2.0** | **10.2±2.3** | **33.0±1.3** | 54.0±1.4 |
| r=80 | PCA | 9.4 | 6.1 | 39.4±1.8 | 39.4±1.8 | 46.0±1.1 | 69.4±3.9 |
|  | LMNN | **1.6±0.1** | 3.2 | 51.1±2.4 | 36.5±2.8 | 43.4±0.9 | 60.3±0.8 |
|  | M²-LMNN | **1.6±0.0** | **1.8** | 51.1±2.4 | 35.9±2.7 | 43.4±1.0 | 54.4±1.5 |
|  | GB-LMNN | **1.6±0.1** | 2.4 | 50.0±1.9 | 27.3±3.5 | 41.1±1.2 | 54.1±1.3 |
|  | $\chi^2$-LMNN | - | - | **33.3±2.0** | **8.3±1.6** | **29.5±1.2** | **51.1±2.3** |

Table 2: $k$NN classification error (in %, $\pm$ standard error where applicable) with dimensionality reduction to output dimensionality $r$. Best results up to one standard error in **bold**.

indicating that convex algorithms (LMNN) as initialization for more expressive non-convex methods can be a winning combination.

The strong results obtained with $\chi^2$-LMNN show that the incorporation of data-specific constraints can be highly beneficial—indicating that there is great potential for future research in specialized metric learning algorithms for specific data types. Further, the ability of $\chi^2$-LMNN to reduce the dimensionality of data sampled from probability simplexes is highly encouraging and might lead to interesting applications in computer vision and other fields, where histogram data is ubiquitous. Here, it might be possible to reduce the running time of time critical algorithms drastically by shrinking the data dimensionality, while strictly maintaining its histogram properties.

The high consistency with which GB-LMNN obtains state-of-the-art results across diverse data sets is highly encouraging. In fact, the use of ensembles of CART trees [4] not only inherits all positive aspects of gradient boosting (robustness, speed and insensitivity to hyper-parameters) but is also a natural match for metric learning. Each tree splits the space into different regions and in contrast to prior work [30], this splitting is fully automated, results in new (discriminatively learned) Euclidean representations of the data and gives rise to well-defined pseudo-metrics.

# 8   Acknowledgements

KQW, DK and ST would like to thank NIH for their support through grant U01 1U01NS073457-01 and NSF for grants 1149882 and 1137211. FS would like to thank DARPA for its support with grant D11AP00278 and ONR for grant N00014-12-1-0066. GL was supported in part by the NSF under Grants CCF-0830535 and IIS-1054960, and by the Sloan Foundation. DK would also like to thank the McDonnell International Scholars Academy for their support.

## Footnotes

[1]The set of all possible matrices $f(\mathbf{A})$ is slightly more restricted than $\mathcal{P}$, as it reaches zero entries only in the limit. However, given finite computational precision, this does not seem to be a problem in practice.

[2]Here, we set the step-size, a common hyper-parameter across all variations of LMNN, to $\alpha = 0.01$.

[3]In the case of *ISOLET*, which consists of audio signals of spoken letters by different individuals, the hold-out set consisted of one speaker.

# References

[1] B. Babenko, S. Branson, and S. Belongie. Similarity metrics for categorization: from monolithic to category specific. In *ICCV '09*, pages 293–300. IEEE, 2009.

[2] A. Beck and M. Teboulle. Mirror descent and nonlinear projected subgradient methods for convex optimization. *Operations Research Letters*, 31(3):167–175, 2003.

[3] D.M. Blei, A.Y. Ng, and M.I. Jordan. Latent dirichlet allocation. *The Journal of Machine Learning Research*, 3:993–1022, 2003.

[4] L. Breiman. *Classification and regression trees*. Chapman & Hall/CRC, 1984.

[5] R. Chatpatanasiri, T. Korsrilabutr, P. Tangchanachaianan, and B. Kijsirikul. A new kernelization framework for mahalanobis distance learning algorithms. *Neurocomputing*, 73(10-12):1570–1579, 2010.

[6] S. Chopra, R. Hadsell, and Y. LeCun. Learning a similarity metric discriminatively, with application to face verification. In *CVPR '05*, pages 539–546. IEEE, 2005.

[7] T. Cover and P. Hart. Nearest neighbor pattern classification. *IEEE Transactions on Information Theory*, 13(1):21–27, 1967.

[8] O.G. Cula and K.J. Dana. 3D texture recognition using bidirectional feature histograms. *International Journal of Computer Vision*, 59(1):33–60, 2004.

[9] M. Cuturi and D. Avis. Ground metric learning. *arXiv preprint, arXiv:1110.2306*, 2011.

[10] J.V. Davis, B. Kulis, P. Jain, S. Sra, and I.S. Dhillon. Information-theoretic metric learning. In *ICML '07*, pages 209–216. ACM, 2007.

[11] J.H. Friedman. Greedy function approximation: a gradient boosting machine. *Annals of Statistics*, pages 1189–1232, 2001.

[12] C. Galleguillos, B. McFee, S. Belongie, and G. Lanckriet. Multi-class object localization by combining local contextual interactions. *CVPR '10*, pages 113–120, 2010.

[13] A. Globerson and S. Roweis. Metric learning by collapsing classes. In *NIPS '06*, pages 451–458. MIT Press, 2006.

[14] A. Globerson and S. Roweis. Visualizing pairwise similarity via semidefinite programming. In *AISTATS '07*, pages 139–146, 2007.

[15] J. Goldberger, S. Roweis, G. Hinton, and R. Salakhutdinov. Neighbourhood components analysis. In *NIPS '05*, pages 513–520. MIT Press, 2005.

[16] J. Hafner, H.S. Sawhney, W. Equitz, M. Flickner, and W. Niblack. Efficient color histogram indexing for quadratic form distance functions. *Pattern Analysis and Machine Intelligence, IEEE Transactions on*, 17(7):729–736, 1995.

[17] M. Hoffman, D. Blei, and P. Cook. Easy as CBA: A simple probabilistic model for tagging music. In *ISMIR '09*, pages 369–374, 2009.

[18] P. Jain, B. Kulis, J.V. Davis, and I.S. Dhillon. Metric and kernel learning using a linear transformation. *Journal of Machine Learning Research*, 13:519–547, 03 2012.

[19] A.M. Mood, F.A. Graybill, and D.C. Boes. *Introduction in the theory of statistics*. McGraw-Hill International Book Company, 1963.

[20] O. Pele and M. Werman. The quadratic-chi histogram distance family. *ECCV '10*, pages 749–762, 2010.

[21] Y. Rubner, C. Tomasi, and L.J. Guibas. The earth mover's distance as a metric for image retrieval. *International Journal of Computer Vision*, 40(2):99–121, 2000.

[22] K. Saenko, B. Kulis, M. Fritz, and T. Darrell. Adapting visual category models to new domains. *Computer Vision–ECCV 2010*, pages 213–226, 2010.

[23] G. Shakhnarovich. *Learning task-specific similarity*. PhD thesis, MIT, 2005.

[24] N. Shental, T. Hertz, D. Weinshall, and M. Pavel. Adjustment learning and relevant component analysis. In *ECCV '02*, volume 4, pages 776–792. Springer-Verlag, 2002.

[25] M. Stricker and M. Orengo. Similarity of color images. In *Storage and Retrieval for Image and Video Databases*, volume 2420, pages 381–392, 1995.

[26] L. Torresani and K. Lee. Large margin component analysis. *NIPS '07*, pages 1385–1392, 2007.

[27] T. Tuytelaars and K. Mikolajczyk. Local invariant feature detectors: a survey. *Foundations and Trends® in Computer Graphics and Vision*, 3(3):177–280, 2008.

[28] S. Tyree, K.Q. Weinberger, K. Agrawal, and J. Paykin. Parallel boosted regression trees for web search ranking. In *WWW '11*, pages 387–396. ACM, 2011.

[29] M. Varma and A. Zisserman. A statistical approach to material classification using image patch exemplars. *Pattern Analysis and Machine Intelligence, IEEE Transactions on*, 31(11):2032–2047, 2009.

[30] K.Q. Weinberger and L.K. Saul. Fast solvers and efficient implementations for distance metric learning. In *ICML '08*, pages 1160–1167. ACM, 2008.

[31] K.Q. Weinberger and L.K. Saul. Distance metric learning for large margin nearest neighbor classification. *The Journal of Machine Learning Research*, 10:207–244, 2009.

[32] E. P. Xing, A. Y. Ng, M. I. Jordan, and S. Russell. Distance metric learning, with application to clustering with side-information. In *NIPS '02*, pages 505–512. MIT Press, 2002.

[33] P.N. Yianilos. Data structures and algorithms for nearest neighbor search in general metric spaces. In *ACM-SIAM Symposium on Discrete Algorithms '93*, pages 311–321, 1993.

